# Linear Dependent Dimensionality Reduction

**Nathan Srebro**     **Tommi Jaakkola**
Department of Electrical Engineering and Computer Science
Massachusetts Institute of Technology
Cambridge, MA 02139
`nati@mit.edu,tommi@ai.mit.edu`

## Abstract

We formulate linear dimensionality reduction as a semi-parametric estimation problem, enabling us to study its asymptotic behavior. We generalize the problem beyond additive Gaussian noise to (unknown) non-Gaussian additive noise, and to unbiased non-additive models.

## 1  Introduction

Factor models are often natural in the analysis of multi-dimensional data. The underlying premise of such models is that the important aspects of the data can be captured via a low-dimensional representation ("factor space"). The low-dimensional representation may be useful for lossy compression as in typical applications of PCA, for signal reconstruction as in factor analysis or non-negative matrix factorization [1], for understanding the signal structure [2], or for prediction as in applying SVD for collaborative filtering [3]. In many situations, including collaborative filtering and structure exploration, the "important" aspects of the data are the dependencies between different attributes. For example, in collaborative filtering we rely on a representation that summarizes the dependencies among user preferences. More generally, we seek to identify a low-dimensional space that captures the *dependent* aspects of the data, and separate them from *independent* variations. Our goal is to relax restrictions on the form of each of these components, such as Gaussianity, additivity and linearity, while maintaining a principled rigorous framework that allows analysis of the methods.

We begin by studying the probabilistic formulations of the problem, focusing on the assumptions that are made about the dependent, low-rank "signal" and independent "noise" distributions. We consider a general semi-parametric formulation that emphasizes what is being estimated and allows us to discuss asymptotic behavior (Section 2). We then study the standard (PCA) approach, show that it is appropriate for additive i.i.d. noise (Section 3), and present a generic estimator that is appropriate also for unbiased non-additive models (Section 4). In Section 5 we confront the non-Gaussianity directly, develop maximum-likelihood estimators in the presence of Gaussian mixture additive noise, and show that the consistency of such maximum-likelihood estimators should not be taken for granted.

## 2 Dependent Dimensionality Reduction

Our starting point is the problem of identifying linear dependencies in the presence of independent identically distributed Gaussian noise. In this formulation, we observe a data matrix $Y \in \Re^{n \times d}$ which we assume was generated as $\mathbf{Y} = X + \mathbf{Z}$, where the dependent, low-dimensional component $X \in \Re^{n \times d}$ (the "signal") is a matrix of rank $k$ and the independent component $\mathbf{Z}$ (the "noise") is i.i.d. zero-mean Gaussian with variance $\sigma^2$. We can write down the log-likelihood of $X$ as $\frac{-1}{\sigma^2}|Y - X|_{\text{Fro}} + \text{Const}$ (where $||_{\text{Fro}}$ is the Frobenius, or sum-squared, norm) and conclude that, regardless of the variance $\sigma^2$, the maximum-likelihood estimator of $X$ is the rank-$k$ matrix minimizing the Frobenius distance. It is given by the leading components of the singular value decomposition of $Y$.[1]

Although the above formulation is perfectly valid, there is something displeasing about it. We view the entire matrix $X$ as parameters, and estimate them according to a single observation $Y$. The number of parameters is linear in the data, and even with more data, we cannot hope to estimate the parameters (entries in $X$) beyond a fixed precision. What we *can* estimate with more data rows is the rank-$k$ row-space of $X$. Consider the factorization $X = UV'$, where $V' \in \Re^{k \times d}$ spans this "signal space". The dependencies of each row $\mathbf{y}$ of $\mathbf{Y}$ are captured by a row $u$ of $U$, which, through the parameters $V$ and $\sigma$ specifies how each entry $\mathbf{y}_i$ is generated *independently* given $u$.[2]

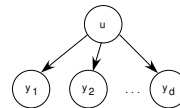

A standard parametric analysis of the model would view $\mathbf{u}$ as a random vector (rather than parameters) and impose some, possibly parametric, distribution over it (interestingly, if $\mathbf{u}$ is Gaussian, the maximum-likelihood reconstruction is the same Frobenius low-rank approximation [4]). However, in the analysis we started with, we did not make any assumptions about the distribution of $\mathbf{u}$, beyond its dimensionality. The model class is then non-parametric, yet we still desire, and are able, to estimate a parametric aspect of the model: The estimator can be seen as a ML estimator for the signal subspace, where the distribution over $\mathbf{u}$ is unconstrained nuisance.

Although we did not impose any form on the distribution $\mathbf{u}$, we did impose a strict form on the conditional distributions $\mathbf{y}_i|\mathbf{u}$: we required them to be Gaussian with fixed variance $\sigma^2$ and mean $\mathbf{u}V_i'$. We would like to relax these requirements, and require only that $\mathbf{y}|\mathbf{u}$ be a product distribution, i.e. that its coordinates $\mathbf{y}_i|\mathbf{u}$ be (conditionally) independent. Since $\mathbf{u}$ is continuous, we cannot expect to forego all restrictions on $\mathbf{y}_i|\mathbf{u}_i$, but we can expect to set up a semi-parametric problem in which $\mathbf{y}|\mathbf{u}$ may lie in an infinite dimensional family of distributions, and is not strictly parameterized.

Relaxing the Gaussianity leads to linear additive models $\mathbf{y} = \mathbf{u}V' + \mathbf{z}$, with $\mathbf{z}$ independent of $\mathbf{u}$, but not necessarily Gaussian. Further relaxing the additivity is appropriate, e.g., when the noise has a multiplicative component, or when the features of $\mathbf{y}$ are not real numbers. These types of models, with a *known* distribution $\mathbf{y}_i|\mathbf{x}_i$, have been suggested for classification using logistic loss [5], when $\mathbf{y}_i|\mathbf{x}_i$ forms an exponential family [6], and in a more abstract framework [7]. Relaxing the linearity assumption $\mathbf{x} = \mathbf{u}V'$ is also appropriate in many situations. Fitting a non-linear manifold by minimizing the sum-squared distance can be seen as a ML estimator for $\mathbf{y}|\mathbf{u} = g(\mathbf{u}) + \mathbf{z}$, where $\mathbf{z}$ is i.i.d. Gaussian and $g : \Re^k \rightarrow \Re^d$ specifies some smooth manifold. Combining these ideas leads us to discuss the conditional distributions $\mathbf{y}_i|g_i(\mathbf{u})$, or $\mathbf{y}_i|\mathbf{u}$ directly.

In this paper we take our first steps is studying this problem, and relaxing restrictions on

$\mathbf{y}|\mathbf{u}$. We continue to assume a linear model $\mathbf{x} = \mathbf{u}V'$ and limit ourselves to additive noise models and unbiased models in which $\mathbf{E}\left[\mathbf{y}|\mathbf{x}\right] = \mathbf{x}$. We study the estimation of the rank-$k$ signal space in which $\mathbf{x}$ resides, based on a sample of $n$ independent observations of $\mathbf{y}$ (forming the rows of $\mathbf{Y}$), where the distribution on $\mathbf{u}$ is unconstrained nuisance.

In order to study estimators for a subspace, we must be able to compare two subspaces. A natural way of doing so is through the *canonical angles* between them [8]. Define the angle between a vector $v_1$ and a subspace $\mathcal{V}_2$ to be the minimal angle between $v_1$ and any $v_2 \in \mathcal{V}_2$. The largest canonical angle between two subspaces is then the maximal angle between a vector in $v_1 \in \mathcal{V}_1$ and the subspace $\mathcal{V}_2$. The second largest angle is the maximum over all vectors orthogonal to the $v_1$, and so on. It is convenient to think of a subspace in terms of the matrix whose columns span it. Computationally, if the columns of $V_1$ and $V_2$ form orthonormal bases of $\mathcal{V}_1$ and $\mathcal{V}_2$, then the cosines of the canonical angles between $\mathcal{V}_1$ and $\mathcal{V}_2$ are given by the singular values of $V_1'V_2$. Throughout the presentation, we will slightly overload notation and use a matrix to denote also its column subspace. In particular, we will denote by $V_0$ the true signal subspace, i.e. such that $\mathbf{x} = \mathbf{u}V_0{}'$.

## 3  The $L_2$ Estimator

We first consider the "standard" approach to low-rank approximation—minimizing the sum squared error.[3] This is the ML estimator when the noise is i.i.d. Gaussian. But the $L_2$ estimator is appropriate also in a more general setting. We will show that the $L_2$ estimator is consistent for any i.i.d. additive noise with finite variance (as we will see later on, this is more than can be said for some ML estimators).

The $L_2$ estimator of the signal subspace is the subspace spanned by the leading eigenvectors of the empirical covariance matrix $\hat{\Lambda}_n$ of $\mathbf{y}$, which is a consistent estimator of the true covariance matrix $\Lambda_Y$, which in turn is the sum of the covariance matrices of $\mathbf{x}$ and $\mathbf{z}$, where $\Lambda_X$ is of rank exactly[4] $k$, and if $\mathbf{z}$ is i.i.d., $\Lambda_Z = \sigma^2 I$.

Let $s_1 \geq s_2 \geq \cdots \geq s_k > 0$ be the non-zero eigenvalues of $\Lambda_{\mathbf{x}}$. Since $\mathbf{z}$ has variance exactly $\sigma^2$ in any direction, the principal directions of variation are not affected by it, and the eigenvalues of $\Lambda_Y$ are exactly $s_1 + \sigma^2, \ldots, s_k + \sigma^2, \sigma^2, \ldots, \sigma^2$, with the leading $k$ eigenvectors being the eigenvectors of $\Lambda_X$. This ensures an eigenvalue gap of $s_k > 0$ between the invariant subspace of $\Lambda_Y$ spanned by the eigenvectors of $\Lambda_X$ and its complement, and we can bound the norm of the canonical sines between $V_0$ and the leading $k$ eigenvectors of $\hat{\Lambda}_n$ by $\frac{|\hat{\Lambda}_n - \Lambda_Y|}{s_k}$ [8]. Since $|\hat{\Lambda}_n - \Lambda_Y| \to 0$ a.s., we conclude that the estimator is consistent.

## 4  The Variance-Ignoring Estimator

We turn to additive noise with independent, but not identically distributed, coordinates. If the noise variances are known, the ML estimator corresponds to minimizing the column-weighted (inversely proportional to the variances) Frobenius norm of $Y - X$, and can be calculated from the leading eigenvectors of a scaled empirical covariance matrix [9]. If the variances are not known, e.g. when the scale of different coordinates is not known, there is no ML estimator: at least $k$ coordinates of each $y$ can always be exactly matched, and so the likelihood is unbounded when up to $k$ variances approach zero.

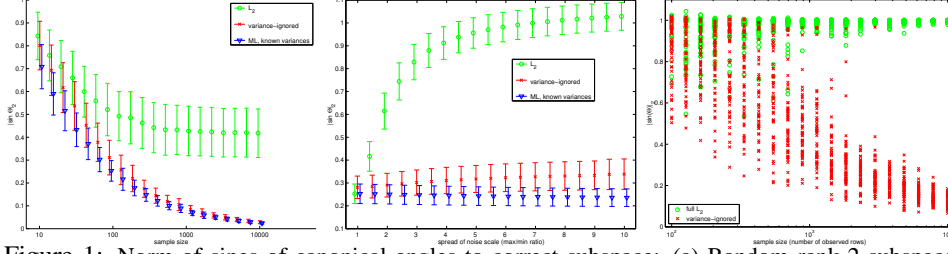

Figure 1: Norm of sines of canonical angles to correct subspace: (a) Random rank-2 subspaces in $\Re^{10}$. Gaussian noise of different scales in different coordinates— between 0.17 and 1.7 signal strength. (b) Random rank-2 subspaces in $\Re^{10}$, 500 sample rows, and Gaussian noise with varying distortion (mean over 200 simulations, bars are one standard deviations tall) (c) Observations are exponentially distributed with means in rank-2 subspace $\left(\begin{smallmatrix} 1 & 1 & 1 & 1 & 1 & 1 & 1 & 1 & 1 & 1 \\ 1 & 0 & 1 & 0 & 1 & 0 & 1 & 0 & 1 & 0 \end{smallmatrix}\right)'$.

The $L_2$ estimator is not satisfactory in this scenario. The covariance matrix $\Lambda_Z$ is still diagonal, but is no longer a scaled identity. The additional variance introduced by the noise is different in different directions, and these differences may overwhelm the "signal" variance along $V_0$, biasing the leading eigenvectors of $\Lambda_Y$, and thus the limit of the $L_2$ estimator, toward axes with high "noise" variance. The fact that this variability is independent of the variability in other coordinates is ignored, and the $L_2$ estimator is asymptotically biased.

Instead of recovering the directions of greatest variability, we recover the covariance structure directly. In the limit, $\hat{\Lambda}_n \to \Lambda_Y = \Lambda_X + \Lambda_Z$, a sum of a rank-$k$ matrix and a diagonal matrix. In particular, the non-diagonal entries of $\hat{\Lambda}_n$ approach those of $\Lambda_X$. We can thus seek a rank-$k$ matrix $\hat{\Lambda}_X$ approximating $\hat{\Lambda}_n$, e.g. in a sum-squared sense, except on the diagonal. This is a (zero-one) *weighted* low-rank approximation problem. We optimize $\hat{\Lambda}_X$ by iteratively seeking a rank-$k$ approximation of $\hat{\Lambda}_n$ with diagonal entries filled in from the last iterate of $\hat{\Lambda}_X$ (this can be viewed as an EM procedure [5]). The row-space of the resulting $\hat{\Lambda}_X$ is then an estimator for the signal subspace. Note that the $L_2$ estimator is the row-space of the rank-$k$ matrix minimizing the *unweighted* sum-squared distance to $\hat{\Lambda}_n$.

Figures 1(a,b) demonstrate this variance-ignoring estimator on simulated data with non-identical Gaussian noise. The estimator reconstructs the signal-space almost as well as the ML estimator, even though it does not have access to the true noise variance.

Discussing consistency in the presence of non-identical noise with unknown variances is problematic, since the signal subspace is not necessarily identifiable. For example, the combined covariance matrix $\Lambda_Y = \left(\begin{smallmatrix} 2 & 1 \\ 1 & 2 \end{smallmatrix}\right)$ can arise from a rank-one signal covariance $\Lambda_X = \left(\begin{smallmatrix} a & 1 \\ 1 & 1/a \end{smallmatrix}\right)$ for any $\frac{1}{2} \le a \le 2$, each corresponding to a different signal subspace. Counting the number of parameters and constraints suggests identifiability when $k < d - \frac{\sqrt{8d+1}-1}{2}$, but this is by no means a precise guarantee. Anderson and Rubin [10] present several conditions on $\Lambda_X$ which are sufficient for identifiability but require $k < \lfloor \frac{d}{2} \rfloor$, and other weaker conditions which are necessary.

**Non-Additive Noise** The above estimation method is also useful in a less straightforward situation. Until now we have considered only additive noise, in which the distribution of $\mathbf{y}_i - \mathbf{x}_i$ was independent of $\mathbf{x}_i$. We will now relax this restriction and allow more general conditional distributions $\mathbf{y}_i | \mathbf{x}_i$, requiring only that $\mathbf{E}\left[\mathbf{y}_i | \mathbf{x}_i\right] = \mathbf{x}_i$. With this requirement, together with the structural constraint ($\mathbf{y}_i$ independent given $\mathbf{x}$), for any $i \ne j$:

$$\text{Cov}\left[\mathbf{y}_i, \mathbf{y}_j\right] = \mathbf{E}\left[\mathbf{y}_i \mathbf{y}_j\right] - \mathbf{E}\left[\mathbf{y}_i\right]\mathbf{E}\left[\mathbf{y}_j\right] = \mathbf{E}\left[\mathbf{E}\left[\mathbf{y}_i \mathbf{y}_j | \mathbf{x}\right]\right] - \mathbf{E}\left[\mathbf{E}\left[\mathbf{y}_i | \mathbf{x}\right]\right]\mathbf{E}\left[\mathbf{E}\left[\mathbf{y}_j | \mathbf{x}\right]\right]$$
$$= \mathbf{E}\left[\mathbf{E}\left[\mathbf{y}_i | \mathbf{x}\right]\mathbf{E}\left[\mathbf{y}_j | \mathbf{x}\right]\right] - \mathbf{E}\left[\mathbf{x}_i\right]\mathbf{E}\left[\mathbf{x}_j\right] = \mathbf{E}\left[\mathbf{x}_i \mathbf{x}_j\right] - \mathbf{E}\left[\mathbf{x}_i\right]\mathbf{E}\left[\mathbf{x}_j\right] = \text{Cov}\left[\mathbf{x}_i, \mathbf{x}_j\right].$$

As in the non-identical additive noise case, $\Lambda_Y$ agrees with $\Lambda_X$ except on the diagonal. Even if $\mathbf{y}_i|\mathbf{x}_i$ is identically conditionally distributed for all $i$, the difference $\Lambda_Y - \Lambda_X$ is *not* in general a scaled identity: $\mathrm{Var}\left[\mathbf{y}_i\right] = \mathbf{E}\left[\mathbf{E}\left[\mathbf{y}_i^2|\mathbf{x}_i\right] - \mathbf{E}\left[\mathbf{y}_i|\mathbf{x}_i\right]^2\right] + \mathbf{E}\left[\mathbf{E}\left[\mathbf{y}_i|\mathbf{x}_i\right]^2\right] - \mathbf{E}\left[\mathbf{y}_i\right]^2 = \mathbf{E}\left[\mathrm{Var}\left[\mathbf{y}_i|\mathbf{x}_i\right]\right] + \mathrm{Var}\left[\mathbf{x}_i\right]$. Unlike the additive noise case, the variance of $\mathbf{y}_i|\mathbf{x}_i$ depends on $\mathbf{x}_i$, and so its expectation depends on the distribution of $\mathbf{x}_i$.

These observations suggest using the variance-ignoring estimator. Figure 1(c) demonstrates how such an estimator succeeds in reconstruction when $\mathbf{y}_i|\mathbf{x}_i$ is exponentially distributed with mean $\mathbf{x}_i$, even though the standard $L_2$ estimator is not applicable. We cannot guarantee consistency because the decomposition of the covariance matrix might not be unique, but when $k < \left\lfloor \frac{d}{2} \right\rfloor$ this is not likely to happen. Note that if the conditional distribution $\mathbf{y}|\mathbf{x}$ is known, even if the decomposition is not unique, the correct signal covariance might be identifiable based on the relationship between the signal marginals and the expected conditional variance of of $\mathbf{y}|\mathbf{x}$, but this is not captured by the variance-ignoring estimator.

## 5 Low Rank Approximation with a Gaussian Mixture Noise Model

We return to additive noise, but seeking better estimation with limited data, we confront non-Gaussian noise distributions directly: we would like to find the maximum-likelihood $X$ when $Y = X + \mathbf{Z}$, and $\mathbf{Z}_{ij}$ are distributed according to a Gaussian mixture: $p_Z(z_{ij}) = \sum_{c=1}^{m} p_c (2\pi\sigma_c^2)^{1/2} \exp((z_{ij} - \mu_c)^2/(2\sigma_c^2))$.

To do so, we introduce latent variables $\mathbf{C}_{ij}$ specifying the mixture component of the noise at $Y_{ij}$, and solve the problem using EM. In the **E**xpectation step, we compute the posterior probabilities $\Pr\left(\mathbf{C}_{ij}|Y_{ij}; X\right)$ based on the current low-rank parameter matrix $X$. In the **M**aximization step we need to find the low-rank matrix $X$ that maximizes the posterior expected log-likelihood:

$$\mathbf{E}_{\mathbf{C}|Y}\left[\log \Pr\left(Y = X + \mathbf{Z}|\mathbf{C}; X\right)\right] = -\sum_{ij}\sum_{c} \frac{\Pr(\mathbf{C}_{ij}=c)|Y_{ij}}{2\sigma_c^2}(X_{ij} - (Y_{ij} + \mu_c))^2 + \text{Const}$$

$$= -\frac{1}{2}\sum_{ij} W_{ij}\left(X_{ij} - A_{ij}\right)^2 + \text{Const} \tag{1}$$

$$\text{where} \quad W_{ij} = \sum_c \frac{\Pr(\mathbf{C}_{ij}=c)|Y_{ij}}{\sigma_c^2} \qquad A_{ij} = Y_{ij} + \sum_c \frac{\Pr(\mathbf{C}_{ij}=c)|Y_{ij}\mu_c}{\sigma_c^2 W_{ij}}$$

This is a *weighted* Frobenius low-rank approximation (WLRA) problem. Equipped with a WLRA optimization method [5], we can now perform EM iteration in order to find the matrix $X$ maximizing the likelihood of the observed matrix $Y$. At each **M** step it is enough to perform a single WLRA optimization iteration, which is guaranteed to improve the WLRA objective, and so also the likelihood. The method can be augmented to handle an *unknown* Gaussian mixture, by introducing an optimization of the mixture parameters at each **M** iteration.

**Experiments with GSMs** We report here initial experiments with ML estimation using bounded Gaussian scale mixtures [11], i.e. a mixture of Gaussians with zero mean, and variance bounded from bellow. Gaussian scale mixtures (GSMs) are a rich class of symmetric distributions, which include non-log-concave, and heavy tailed distributions. We investigated two noise distributions: a 'Gaussian with outliers' distribution formed as a mixture of two zero-mean Gaussians with widely varying variances; and a Laplace distribution $p(z) \propto e^{-|z|}$, which is an infinite scale mixture of Gaussians. Figures 2(a,b) show the quality of reconstruction of the $L_2$ estimator and the ML bounded GSM estimator, for these two noise distributions, for a fixed sample size of 300 rows, under varying

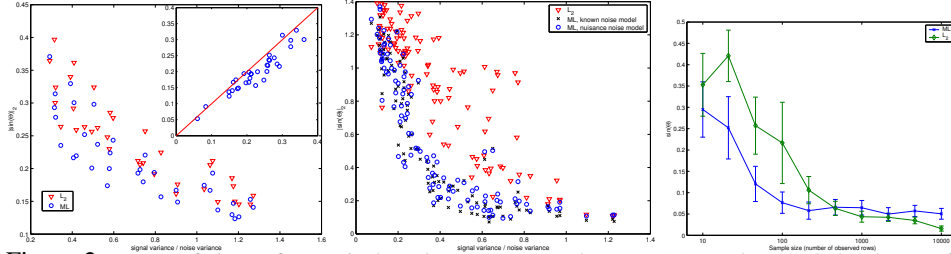

Figure 2: Norm of sines of canonical angles to correct subspace: (a) Random rank-3 subspace in $\Re^{10}$ with Laplace noise. Insert: sine norm of ML est. plotted against sine norm of $L_2$ est. (b) Random rank-2 subspace in $\Re^{10}$ with $0.99\mathcal{N}(0,1) + 0.01\mathcal{N}(0,100)$ noise. (c) $\mathrm{span}(2,1,1)' \subset \Re^3$ with $0.9\mathcal{N}(0,1) + 0.1\mathcal{N}(0,25)$ noise. The ML estimator converges to $(2.34, 1, 1)$. Bars are one standard deviation tall.

signal strengths. We allowed ten Gaussian components, and did not observe any significant change in the estimator when the number of components increases.

The ML estimator is overall more accurate than the $L_2$ estimator—it succeeds in reliably reconstructing the low-rank signal for signals which are approximately three times weaker than those necessary for reliable reconstruction using the $L_2$ estimator. The improvement in performance is not as dramatic, but still noticeable, for Laplace noise.

**Comparison with Newton's Methods** Confronted with a general additive noise distribution, the approach presented here would be to rewrite, or approximate, it as a Gaussian mixture and use WLRA in order to learn $X$ using EM. A different approach is to considering the second order Taylor expansions of the log-likelihood, with respect to the entries of $X$, and iteratively maximize them using WLRA [5, 7]. Such an approach requires calculating the first and second derivatives of the density. If the density is not specified analytically, or is unknown, these quantities need to be estimated. But beyond these issues, which can be overcome, lies the major problem of Newton's method: the noise density must be strictly log-concave and differentiable. If the distribution is not log-concave, the quadratic expansion of the log-likelihood will be unbounded and will not admit an optimum. Attempting to ignore this fact, and for example "optimizing" $U$ given $V$ using the equations derived for non-negative weights would actually drive us towards a saddle-point rather then a local optimum. The non-concavity does not only mean that we are not guaranteed a global optimum (which we are not guaranteed in any case, due to the non-convexity of the low-rank requirement)— it does not yield even local improvements. On the other hand, approximating the distribution as a Gaussians mixture and using the EM method, might still get stuck in local minima, but is at least guaranteed local improvement.

Limiting ourselves to only log-concave distributions is a rather strong limitation, as it precludes, for example, all heavy-tailed distributions. Consider even the "balanced tail" Laplace distribution $p(z) \propto e^{-|z|}$. Since the log-density is piecewise linear, a quadratic approximation of it is a line, which of course does not attain a minimum value.

**Consistency** Despite the gains in reconstruction presented above, the ML estimator may suffer from an asymptotic bias, making it inferior to the $L_2$ estimator on large samples. We study the asymptotic limit of the ML estimator, for a known product distribution $p$. We first establish a necessary and sufficient condition for consistency of the estimator.

The ML estimator is the minimizer of the empirical mean of the random function $\Phi(V) = \min_u(-\log p(\mathbf{y} - uV'))$. When the number of samples increase, the empirical means converge to the true means, and if $\mathbf{E}\left[\Phi(V_1)\right] < \mathbf{E}\left[\Phi(V_2)\right]$, then with probability approaching

one $V_2$ will not minimize $\hat{\mathbf{E}}\left[\Phi(V)\right]$. For the ML estimator to be consistent, $\mathbf{E}\left[\Phi(V)\right]$ *must* be minimized by $V_0$, establishing a necessary condition for consistency.

The sufficiency of this condition rests on the *uniform* convergence of $\{\hat{\mathbf{E}}\left[\Phi(V)\right]\}$, which does not generally exist, or at least on uniform *divergence* from $\mathbf{E}\left[\Phi(V_0)\right]$. It should be noted that the issue here is whether the ML estimator at all converges, since if it does converge, it must converge to the minimizer of $\mathbf{E}\left[\Phi(V)\right]$. Such convergence can be demonstrated at least in the special case when the marginal noise density $p(z_i)$ is continuous, strictly positive, and has finite variance and differential entropy. Under these conditions, the ML estimator is consistent if and only if $V_0$ is the unique minimizer of $\mathbf{E}\left[\Phi(V)\right]$.

When discussing $\mathbf{E}\left[\Phi(V)\right]$, the expectation is with respect to the noise distribution *and* the signal distribution. This is not quite satisfactory, as we would like results which are independent of the signal distribution, beyond the rank of its support. To do so, we must ensure the expectation of $\Phi(V)$ is minimized on $V_0$ for *all* possible signals (and not only in expectation). Denote the objective $\phi(y; V) = \min_u(-\log p(y - uV'))$. For any $x \in \Re^d$, consider $\Psi(V; x) = \mathbf{E}_{\mathbf{z}}\left[\phi(x + \mathbf{z}; V)\right]$, where the expectation is only over the additive noise $\mathbf{z}$. Under the previous conditions guaranteeing the ML estimator converges, it is consistent for any signal distribution if and only if, for all $x \in \Re^d$, $\Psi(V; x)$ is minimized with respect to $V$ exactly when $x \in \mathrm{span} V$.

It will be instructive to first revisit the ML estimator in the presence of i.i.d. Gaussian noise, i.e. the $L_2$ estimator which we already showed is consistent. We will consider the decomposition $y = y_\parallel + y_\perp$ of vectors into their projection onto the subspace $V$, and the residual . Any rotation of $p$ is an isotropic Gaussian, and so $\mathbf{z}_\perp$ and $\mathbf{z}_\parallel$ are independent, and $p(y) = p_\parallel(y_\parallel)p_\perp(y_\perp)$. We can now analyze:

$$\phi(V; y) = \min_u(-\log p_\parallel(y_\parallel + uV') - \log p_\perp(y_\perp)) = -\log p_\parallel(0) + \frac{1}{\sigma^2}|y_\perp|_2 + \mathrm{Const}$$

yielding $\Psi(V; x) \propto \mathbf{E}_{\mathbf{z}_\perp}\left[|x_\perp + \mathbf{z}_\perp|_2\right] + \mathrm{Const}$, which is minimized when $x_\perp = 0$, i.e. $x$ is spanned by $V$. We thus re-derived the consistency of the $L_2$ estimator directly, for the special case in which the noise is indeed Gaussian.

This consistency proof employed a key property of the isotropic Gaussian: rotations of an isotropic Gaussian random variable remain i.i.d. As this property is unique to Gaussian random variables, other ML estimators might not be consistent. In fact, we will shortly see that the ML estimator for a known Laplace noise model is not consistent. To do so, we will note that a necessary condition for consistency, if the density function $p$ is continuous, is that $\Psi(V; 0) = \mathbf{E}\left[\phi(\mathbf{z}; V)\right]$ is constant over all $V$. Otherwise we have $\Psi(V_1; 0) < \Psi(V_2; 0)$ for some $V_1, V_2$, and for small enough $x \in V_2$, $\Psi(V_1; x) < \Psi(V_2; x)$. A non-constant $\Psi(V; 0)$ indicates an a-priori bias towards certain sub-spaces.

The negative log-likelihood of a Laplace distribution, $p(z_i) = \frac{1}{2}e^{-|z_i|}$, is essentially the $L_1$ norm. Consider a rank-one approximation in a two-dimensional space with Laplace noise. For any $V = (1, \alpha)$, $0 \le \alpha \le 1$, and $(z_1, z_2)$, the $L_1$ norm $|z + uV'|_1$ is minimized when $z_1 + u = 0$ yielding $\phi(V; z) = |z_2 - \alpha z_1|$, ignoring a constant term, and $\Psi(V; 0) = \int \int \frac{1}{4}e^{-|z_1|-|z_2|}|z_2 - \alpha z_1|dz_1 dz_2 = \frac{\alpha^2 + \alpha + 1}{\alpha + 1}$, which is monotonic increasing in $\alpha$ in the valid range $[0, 1]$. In particular, $1 = \Psi((1, 0); 0) < \Psi((1, 1); 0) = \frac{3}{2}$ and the estimator is biased towards being axis-aligned.

Figure 2(c) demonstrates such an asymptotic bias empirically. Two-component Gaussian mixture noise was added to rank-one signal in $\Re^3$, and the signal subspace was estimated using an ML estimator with known noise model, and an $L_2$ estimator. For small data sets, the ML estimator is more accurate, but as the number of samples increase, the error of the $L_2$ estimator vanishes, while the ML estimator converges to the wrong subspace.

# 6 Discussion

In many applications few assumptions beyond independence can be made. We formulate the problem of dimensionality reduction as semi-parametric estimation of the low-dimensional signal, or "factor" space, treating the signal distribution as unconstrained nuisance and the noise distribution as constrained nuisance. We present an estimator which is appropriate when the conditional means $\mathbf{E}\left[\mathbf{y}|\mathbf{u}\right]$ lie in a low-dimensional *linear* space, and a maximum-likelihood estimator for additive Gaussian mixture noise.

The variance-ignoring estimator is also applicable when $\mathbf{y}$ can be transformed such that $\mathbf{E}\left[g(\mathbf{y})|\mathbf{u}\right]$ lie in a low-rank linear space, e.g. in log-normal models. If the conditional distribution $\mathbf{y}|\mathbf{x}$ is known, this amount to an unbiased estimator for $\mathbf{x}_i$. When such a transformation is not known, we may wish to consider it as nuisance.

We draw attention to the fact the maximum-likelihood low-rank estimation cannot be taken for granted, and demonstrate that it might not be consistent even for known noise models. The approach employed here can also be used to investigate the consistency of ML estimators with non-additive noise models. Of particular interest are distributions $\mathbf{y}_i|\mathbf{x}_i$ that form exponential families where $\mathbf{x}_i$ are the *natural* parameters [6]. When the *mean* parameters form a low-rank linear subspace, the variance-ignoring estimator is applicable, but when the natural parameters form a linear subspace, the means are in general curved, and there is no unbiased estimator for the natural parameters. Initial investigation reveals that, for example, the ML estimator for a Bernoulli (logistic) conditional distribution is not consistent. The problem of finding a consistent estimator for the linear-subspace of natural parameters when $\mathbf{y}_i|\mathbf{x}_i$ forms an exponential family remains open.

We also leave open the efficiency of the various estimators, and the problem of finding asymptotically efficient estimators, and consistent estimators exhibiting the finite-sample gains of the ML estimator for additive Gaussian mixture noise.

## Footnotes

[1] A mean term is also usually allowed. Incorporating a non-zero mean is straight forward, and in order to simplify derivations, we do not account for it in most of our presentation.

[2] We use uppercase letters to denote matrices, and lowercase letters for vectors, and use bold type to indicate random quantities.

[3] We call this an $L_2$ estimator not because it minimizes the matrix $L_2$-norm $|Y - X|_2$, which it does, but because it minimizes the vector $L_2$-norms $|y - x|_2^2$.

[4] We should also be careful about signals that occupy only a proper subspace of $V_0$, and be satisfied with any rank-$k$ subspace containing the support of $\mathbf{x}$, but for simplicity of presentation we assume this does not happen and $\mathbf{x}$ is of full rank $k$.

# References

[1] Daniel D. Lee and H. Sebastian Seung. Learning the parts of objects by non-negative matrix factorization. *Nature*, 401:788–791, 1999.

[2] Orly Alter, Patrick O. Brown, and David Botstein. Singular value decomposition for genome-wide expression data processing and modeling. *PNAS*, 97(18):10101–10106, 2000.

[3] Yossi Azar, Amos Fiat, Anna R. Karlin, Frank McSherry, and Jared Saia. Spectral analysis of data. In *33rd ACM Symposium on Theory of Computing*, 2001.

[4] M. E. Tipping and C. M. Bishop. Probabilistic principal component analysis. *Journal of the Royal Statistical Society, Series B*, 21(3):611–622, 1999.

[5] Nathan Srebro and Tommi Jaakkola. Weighted low rank approximation. In *20th International Conference on Machine Learning*, 2003.

[6] M. Collins, S. Dasgupta, and R. E. Schapire. A generalization of principal components analysis to the exponential family. In *Advances in Neural Information Processing Systems 14*, 2002.

[7] Geoffrey J. Gordon. Generalized$^2$ linear$^2$ models. In *Advances in Neural Information Processing Systems 15*, 2003.

[8] G. W. Stewart and Ji-guang Sun. *Matrix Perturbation Theory*. Academic Press, Inc, 1990.

[9] Michal Irani and P Anandan. Factorization with uncertainty. In *6th European Conference on Computer Vision*, 2000.

[10] T. W. Anderson and Herman Rubin. Statistical inference in factor analysis. In *Third Berleley Symposium on Mathematical Statistics and Probability*, volume V, pages 111–150, 1956.

[11] M J Wainwright and E P Simoncelli. Scale mixtures of Gaussians and the statistics of natural images. In *Advances in Neural Information Processing Systems 12*, 2000.
